# Heuristics for Ordering Cue Search in Decision Making

**Peter M. Todd**            **Anja Dieckmann**
Center for Adaptive Behavior and Cognition
MPI for Human Development
Lentzeallee 94, 14195 Berlin, Germany
*ptodd@mpib-berlin.mpg.de*            *dieckmann@mpib-berlin.mpg.de*

## Abstract

Simple lexicographic decision heuristics that consider cues one at a time in a particular order and stop searching for cues as soon as a decision can be made have been shown to be both accurate and frugal in their use of information. But much of the simplicity and success of these heuristics comes from using an appropriate cue order. For instance, the Take The Best heuristic uses validity order for cues, which requires considerable computation, potentially undermining the computational advantages of the simple decision mechanism. But many cue orders can achieve good decision performance, and studies of sequential search for data records have proposed a number of simple ordering rules that may be of use in constructing appropriate decision cue orders as well. Here we consider a range of simple cue ordering mechanisms, including tallying, swapping, and move-to-front rules, and show that they can find cue orders that lead to reasonable accuracy and considerable frugality when used with lexicographic decision heuristics.

## 1   One-Reason Decision Making and Ordered Search

How do we know what information to consider when making a decision? Imagine the problem of deciding which of two objects or options is greater along some criterion, such as which of two cities is larger. We may know various facts about each city, such as whether they have a major sports team or a university or airport. To decide between them, we could weight and sum all the cues we know, or we could use a simpler lexicographic rule to look at one cue at a time in a particular order until we find a cue that discriminates between the options and indicates a choice [1]. Such lexicographic rules are used by people in a variety of decision tasks [2]-[4], and have been shown to be both accurate in their inferences and frugal in the amount of information they consider before making a decision. For instance, Gigerenzer and colleagues [5] demonstrated the surprising performance of several decision heuristics that stop information search as soon as one discriminating cue is found; because only that cue is used to make the decision, and no integration of information is involved, they called these heuristics "one-reason" decision mechanisms. Given some set of cues that can be looked up to make the decision, these heuristics differ mainly in the search rule that determines the order in which

the information is searched. But then the question of what information to consider becomes, how are these search orders determined?

Particular cue orders make a difference, as has been shown in research on the Take The Best heuristic (TTB) [6], [7]. TTB consists of three building blocks. (1) Search rule: Search through cues in the order of their validity, a measure of accuracy equal to the proportion of correct decisions made by a cue out of all the times that cue discriminates between pairs of options. (2) Stopping rule: Stop search as soon as one cue is found that discriminates between the two options. (3) Decision rule: Select the option to which the discriminating cue points, that is, the option that has the cue value associated with higher criterion values.

The performance of TTB has been tested on several real-world data sets, ranging from professors' salaries to fish fertility [8], in cross-validation comparisons with other more complex strategies. Across 20 data sets, TTB used on average only a third of the available cues (2.4 out of 7.7), yet still outperformed multiple linear regression in generalization accuracy (71% vs. 68%). The even simpler Minimalist heuristic, which searches through available cues in a random order, was more frugal (using 2.2 cues on average), yet still achieved 65% accuracy. But the fact that the accuracy of Minimalist lagged behind TTB by 6 percentage points indicates that part of the secret of TTB's success lies in its ordered search. Moreover, in laboratory experiments [3], [4], [9], people using lexicographic decision strategies have been shown to employ cue orders based on the cues' validities or a combination of validity and discrimination rate (proportion of decision pairs on which a cue discriminates between the two options).

Thus, the cue order used by a lexicographic decision mechanism can make a considerable difference in accuracy; the same holds true for frugality, as we will see. But constructing an exact validity order, as used by Take The Best, takes considerable information and computation [10]. If there are N known objects to make decisions over, and C cues known for each object, then each of the C cues must be evaluated for whether it discriminates correctly (counting up R right decisions), incorrectly (W wrong decisions), or does not discriminate between each of the N·(N-1)/2 possible object pairs, yielding C·N·(N-1)/2 checks to perform to gather the information needed to compute cue validities ($v = R/(R+W)$) in this domain. But a decision maker typically does not know all of the objects to be decided upon, nor even all the cue values for those objects, ahead of time—is there any simpler way to find an accurate and frugal cue order?

In this paper, we address this question through simulation-based comparison of a variety of simple cue-order-learning rules. Hope comes from two directions: first, there are many cue orders besides the exact validity ordering that can yield good performance; and second, research in computer science has demonstrated the efficacy of a range of simple ordering rules for a closely related search problem. Consequently, we find that simple mechanisms at the cue-order-learning stage can enable simple mechanisms at the decision stage, such as lexicographic one-reason decision heuristics, to perform well.

## 2 Simple approaches to constructing cue search orders

To compare different cue ordering rules, we evaluate the performance of different cue orders when used by a one-reason decision heuristic within a particular well-studied sample domain: large German cities, compared on the criterion of population size using 9 cues ranging from having a university to the presence of an intercity train line [6], [7]. Examining this domain makes it clear that there are many good possible cue orders. When used with one-reason stopping and decision building blocks, the mean accuracy of the 362,880 (9!) cue orders is 70%, equivalent to the performance expected from

Minimalist. The accuracy of the validity order, 74.2%, falls toward the upper end of the accuracy range (62-75.8%), but there are still 7421 cue orders that do better than the validity order. The frugality of the search orders ranges from 2.53 cues per decision to 4.67, with a mean of 3.34 corresponding to using Minimalist; TTB has a frugality of 4.23, implying that most orders are more frugal. Thus, there are many accurate and frugal cue orders that could be found—a satisficing decision maker not requiring optimal performance need only land on one.

An ordering problem of this kind has been studied in computer science for nearly four decades, and can provide us with a set of potential heuristics to test. Consider the case of a set of data records arranged in a list, each of which will be required during a set of retrievals with a particular probability $p_i$. On each retrieval, a key is given (e.g. a record's title) and the list is searched from the front to the end until the desired record, matching that key, is found. The goal is to minimize the mean search time for accessing the records in this list, for which the optimal ordering is in decreasing order of $p_i$. But if these retrieval probabilities are not known ahead of time, how can the list be ordered after each successive retrieval to achieve fast access? This is the problem of self-organizing sequential search [11], [12].

A variety of simple sequential search heuristics have been proposed for this problem, centering on three main approaches: (1) transpose, in which a retrieved record is moved one position closer to the front of the list (i.e., swapping with the record in front of it); (2) move-to-front (MTF), in which a retrieved record is put at the front of the list, and all other records remain in the same relative order; and (3) count, in which a tally is kept of the number of times each record is retrieved, and the list is reordered in decreasing order of this tally after each retrieval. Because count rules require storing additional information, more attention has focused on the memory-free transposition and MTF rules. Analytic and simulation results (reviewed in [12]) have shown that while transposition rules can come closer to the optimal order asymptotically, in the short run MTF rules converge more quickly (as can count rules). This may make MTF (and count) rules more appealing as models of cue order learning by humans facing small numbers of decision trials. Furthermore, MTF rules are more responsive to local structure in the environment (e.g., clumped retrievals over time of a few records), and transposition can result in very poor performance under some circumstances (e.g., when neighboring pairs of "popular" records get trapped at the end of the list by repeatedly swapping places).

It is important to note that there are important differences between the self-organizing sequential search problem and the cue-ordering problem we address here. In particular, when a record is sought that matches a particular key, search proceeds until the correct record is found. In contrast, when a decision is made lexicographically and the list of cues is searched through, there is no one "correct" cue to find—each cue may or may not discriminate (allow a decision to be made). Furthermore, once a discriminating cue *is* found, it may not even make the right decision. Thus, given feedback about whether a decision was right or wrong, a discriminating cue could potentially be moved up or down in the ordered list. This dissociation between making a decision or not (based on the cue discrimination rates), and making a right or wrong decision (based on the cue validities), means that there are two ordering criteria in this problem—frugality and accuracy—as opposed to the single order—search time—for records based on their retrieval probability $p_i$. Because record search time corresponds to cue frugality, the heuristics that work well for the self-organizing sequential search task are likely to produce orders that emphasize frugality (reflecting cue discrimination rates) over accuracy in the cue-ordering task. Nonetheless, these heuristics offer a useful starting point for exploring cue-ordering rules.

## 2.1   The cue-ordering rules

We focus on search order construction processes that are psychologically plausible by being frugal both in terms of information storage and in terms of computation. The decision situation we explore is different from the one assumed by Juslin and Persson [10] who strongly differentiate learning about objects from later making decisions about them. Instead we assume a learning-while-doing situation, consisting of tasks that have to be done repeatedly with feedback after each trial about the adequacy of one's decision. For instance, we can observe on multiple occasions which of two supermarket checkout lines, the one we have chosen or (more likely) another one, is faster, and associate this outcome with cues including the lines' lengths and the ages of their respective cashiers. In such situations, decision makers can learn about the differential usefulness of cues for solving the task via the feedback received over time.

We compare several explicitly defined ordering rules that construct cue orders for use by lexicographic decision mechanisms applied to a particular probabilistic inference task: forced choice paired comparison, in which a decision maker has to infer which of two objects, each described by a set of binary cues, is "bigger" on a criterion—just the task for which TTB was formulated. After an inference has been made, feedback is given about whether a decision was right or wrong. Therefore, the order-learning algorithm has information about which cues were looked up, whether a cue discriminated, and whether a discriminating cue led to the right or wrong decision. The rules we propose differ in which pieces of information they use and how they use them. We classify the learning rules based on their memory requirement—high versus low—and their computational requirements in terms of full or partial reordering (see Table 1).

Table 1: Learning rules classified by memory and computational requirements

| High memory load, complete reordering | High memory load, local reordering | Low memory load, local reordering |
|---|---|---|
| Validity: reorders cues based on their current validity<br><br>Tally: reorders cues by number of correct minus incorrect decisions made so far<br><br>Associative/delta rule: reorders cues by learned association strength | Tally swap: moves cue up (down) one position if it has made a correct (incorrect) decision if its tally of correct minus incorrect decisions is ≥ (≤) than that of next higher (lower) cue | Simple swap: moves cue up one position after correct decision, and down after an incorrect decision<br><br>Move-to-front (2 forms):<br>Take The Last (TTL): moves discriminating cue to front<br><br>TTL-correct: moves cue to front only if it correctly discriminates |

The *validity rule*, a type of count rule, is the most demanding of the rules we consider in terms of both memory requirements and computational complexity. It keeps a count of all discriminations made by a cue so far (in all the times that the cue was looked up) and a separate count of all the correct discriminations. Therefore, memory load is comparatively high. The validity of each cue is determined by dividing its current correct discrimination count by its total discrimination count. Based on these values computed after each decision, the rule reorders the whole set of cues from highest to lowest validity.

The *tally rule* only keeps one count per cue, storing the number of correct decisions made by that cue so far minus the number of incorrect decisions. If a cue discriminates correctly on a given trial, one point is added to its tally, if it leads to an incorrect decision, one point is subtracted. The tally rule is less demanding in terms of memory and computation: Only one count is kept, no division is required.

The *simple swap rule* uses the transposition rather than count approach. This rule has no memory of cue performance other than an ordered list of all cues, and just moves a cue up one position in this list whenever it leads to a correct decision, and down if it leads to an incorrect decision. In other words, a correctly deciding cue swaps positions with its nearest neighbor upwards in the cue order, and an incorrectly deciding cue swaps positions with its nearest neighbor downwards.

The *tally swap rule* is a hybrid of the simple swap rule and the tally rule. It keeps a tally of correct minus incorrect discriminations per cue so far (so memory load is high) but only locally swaps cues: When a cue makes a correct decision and its tally is greater than or equal to that of its upward neighbor, the two cues swap positions. When a cue makes an incorrect decision and its tally is smaller than or equal to that of its downward neighbor, the two cues also swap positions.

We also evaluate two types of move-to-front rules. First, the *Take The Last (TTL)* rule moves the last discriminating cue (that is, whichever cue was found to discriminate for the current decision) to the front of the order. This is equivalent to the Take The Last heuristic [6], [7], which uses a memory of cues that discriminated in the past to determine cue search order for subsequent decisions. Second, *TTL-correct* moves the last discriminating cue to the front of the order only if it correctly discriminated; otherwise, the cue order remains unchanged. This rule thus takes accuracy as well as frugality into account.

Finally, we include an associative learning rule that uses the delta rule to update cue weights according to whether they make correct or incorrect discriminations, and then reorders all cues in decreasing order of this weight after each decision. This corresponds to a simple network with nine input units encoding the difference in cue value between the two objects (A and B) being decided on (i.e., $in_i = -1$ if $cue_i(A)<cue_i(B)$, 1 if $cue_i(A)>cue_i(B)$, and 0 if $cue_i(A)=cue_i(B)$ or $cue_i$ was not checked) and with one output unit whose target value encodes the correct decision ($t = 1$ if criterion(A)>criterion(B), otherwise -1), and with the weights between inputs and output updated according to $\Delta w_i = lr \cdot (t - in_i \cdot w_i) \cdot in_i$ with learning rate $lr = 0.1$. We expect this rule to behave similarly to Oliver's rule initially (moving a cue to the front of the list by giving it the largest weight when weights are small) and to swap later on (moving cues only a short distance once weights are larger).

## 3  Simulation Study of Simple Ordering Rules

To test the performance of these order learning rules, we use the German cities data set [6], [7], consisting of the 83 largest-population German cities (those with more than 100,000 inhabitants), described on 9 cues that give some information about population size. Discrimination rate and validity of the cues are negatively correlated (r = -.47). We present results averaged over 10,000 learning trials for each rule, starting from random initial cue orders. Each trial consisted of 100 decisions between randomly selected decision pairs. For each decision, the current cue order was used to look up cues until a discriminating cue was found, which was used to make the decision (employing a one-reason or lexicographic decision strategy). After each decision, the cue order was updated using the particular order-learning rule. We start by considering the cumulative accuracies (i.e., online or amortized performance—[12]) of the rules, defined as the total percentage of correct decisions made so far at any point in the learning process. The

contrasting measure of offline accuracy—how well the current learned cue order would do if it were applied to the entire test set—will be subsequently reported (see Figure 1).

For all but the move-to-front rules, cumulative accuracies soon rise above that of the Minimalist heuristic (proportion correct = .70) which looks up cues in random order and thus serves as a lower benchmark. However, at least throughout the first 100 decisions, cumulative accuracies stay well below the (offline) accuracy that would be achieved by using TTB for all decisions (proportion correct = .74), looking up cues in the true order of their ecological validities. Except for the move-to-front rules, whose cumulative accuracies are very close to Minimalist (mean proportion correct in 100 decisions: TTL: .701; TTL-correct: .704), all learning rules perform on a surprisingly similar level, with less than one percentage point difference in favor of the most demanding rule (i.e., delta rule: .719) compared to the least (i.e., simple swap: .711; for comparison: tally swap: .715; tally: .716; validity learning rule: .719). Offline accuracies are slightly higher, again with the exception of the move to front rules (TTL: .699; TTL-correct: .702; simple swap: .714; tally swap: .719; tally: .721; validity learning rule: .724; delta rule: .725; see Figure 1). In longer runs (10,000 decisions) the validity learning rule is able to converge on TTB's accuracy, but the tally rule's performance changes little (to .73).

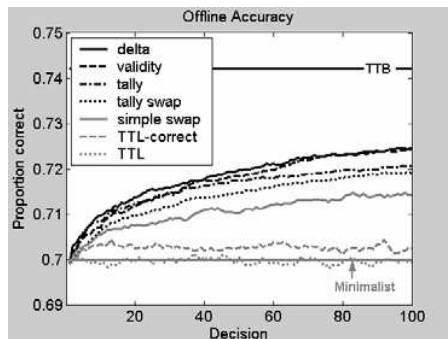 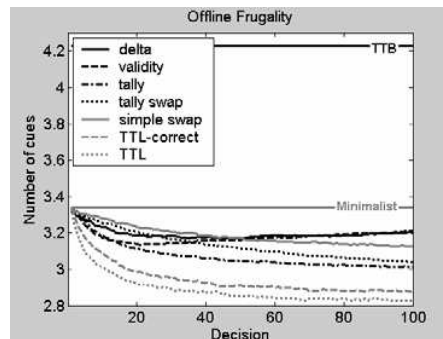

Figure 1: Mean offline accuracy of order learning rules

Figure 2: Mean offline frugality of order learning rules

All learning rules are, however, more *frugal* than TTB, and even more frugal than Minimalist, both in terms of online as well as offline frugality. Let us focus on their offline frugality (see Figure 2): On average, the rules look up fewer cues than Minimalist before reaching a decision. There is little difference between the associative rule, the tallying rules and the swapping rules (mean number of cues looked up in 100 decisions: delta rule: 3.20; validity learning rule: 3.21; tally: 3.01; tally swap: 3.04; simple swap: 3.13). Most frugal are the two move-to front rules (TTL-correct: 2.87; TTL: 2.83).

Consistent with this finding, all of the learning rules lead to cue orders that show positive correlations with the discrimination rate cue order (reaching the following values after 100 decisions: validity learning rule: r = .18; tally: r = .29; tally swap: r = .24; simple swap: r = .18; TTL-correct: r = .48; TTL: r = .56). This means that cues that often lead to discriminations are more likely to end up in the first positions of the order. This is especially true for the move-to-front rules. In contrast, the cue orders resulting from all learning rules but the validity learning rule do not correlate or correlate negatively with the validity cue order, and even the correlations of the cue orders resulting from the validity learning rule after 100 decisions only reach an average r = .12.

But why would the discrimination rates of cues exert more of a pull on cue order than validity, even when the validity learning rule is applied? As mentioned earlier, this is what we would expect for the move-to-front rules, but it was unexpected for the other rules. Part of the explanation comes from the fact that in the city data set we used for the

simulations, validity and discrimination rate of cues are negatively correlated. Having a low discrimination rate means that a cue has little chance to be used and hence to demonstrate its high validity. Whatever learning rule is used, if such a cue is displaced downward to the lower end of the order by other cues, it may have few chances to escape to the higher ranks where it belongs. The problem is that when a decision pair is finally encountered for which that cue would lead to a correct decision, it is unlikely to be checked because other, more discriminating although less valid, cues are looked up before and already bring about a decision. Thus, because one-reason decision making is intertwined with the learning mechanism and so influences which cues can be learned about, what mainly makes a cue come early in the order is producing a high *number* of correct decisions and not so much a high *ratio* of correct discriminations to total discriminations regardless of base rates.

This argument indicates that performance may differ in environments where cue validities and discrimination rates correlate positively. We tested the learning rules on one such data set (r=.52) of mammal species life expectancies, predicted from 9 cues. It also differs from the cities environment with a greater difference between TTB's and Minimalist's performance (6.5 vs. 4 percentage points). In terms of offline accuracy, the validity learning rule now indeed more closely approaches TTB's accuracy after 100 decisions (.773 vs. .782)., The tally rule, in contrast, behaves very much as in the cities environment, reaching an accuracy of .752, halfway between TTB and Minimalist (accuracy =.716). Thus only some learning rules can profit from the positive correlation.

## 4 Discussion

Most of the simpler cue order learning rules we have proposed do not fall far behind a validity learning rule in accuracy, and although the move-to-front rules cannot beat the accuracy achieved if cues were selected randomly, they compensate for this failure by being highly frugal. Interestingly, the rules that do achieve higher accuracy than Minimalist also beat random cue selection in terms of frugality.

On the other hand, all rules, even the delta rule and the validity learning rule, stay below TTB's accuracy across a relatively high number of decisions. But often it is necessary to make good decisions without much experience. Therefore, learning rules should be preferred that quickly lead to orders with good performance. The relatively complex rules with relatively high memory requirement, i.e., the delta and the validity learning rule, but also the tally learning rule, more quickly rise in accuracy compared the rules with lower requirements. Especially the tally rule thus represents a good compromise between cost, correctness and psychological plausibility considerations.

Remember that the rules based on tallies assume full memory of all correct minus incorrect decisions made by a cue so far. But this does not make the rule implausible, at least from a psychological perspective, even though computer scientists were reluctant to adopt such counting approaches because of their extra memory requirements. There is considerable evidence that people are actually very good at remembering the frequencies of events. Hasher and Zacks [13] conclude from a wide range of studies that frequencies are encoded in an automatic way, implying that people are sensitive to this information without intention or special effort. Estes [14] pointed out the role frequencies play in decision making as a shortcut for probabilities. Further, the tally rule and the tally swap rule are comparatively simple, not having to keep track of base rates or perform divisions as does the validity rule. From the other side, the simple swap and move to front rules may not be much simpler, because storing a cue order may be about as demanding as storing a set of tallies. We have run experiments (reported elsewhere) in which indeed the tally swap rule best accounts for people's actual processes of ordering cues.

Our goal in this paper was to explore how well simple cue-ordering rules could work in conjunction with lexicographic decision strategies. This is important because it is

necessary to take into account the set-up costs of a heuristic in addition to its application costs when considering the mechanism's overall simplicity. As the example of the validity search order of TTB shows, what is easy to apply may not necessarily be so easy to set up. But simple rules can also be at work in the construction of a heuristic's building blocks. We have proposed such rules for the construction of one building block, the search order. Simple learning rules inspired by research in computer science can enable a one-reason decision heuristic to perform only slightly worse than if it had full knowledge of cue validities from the very beginning. Giving up the assumption of full a priori knowledge for the slight decrease in accuracy seems like a reasonable bargain: Through the addition of learning rules, one-reason decision heuristics might lose some of their appeal to decision theorists who were surprised by the performance of such simple mechanisms compared to more complex algorithms, but they gain psychological plausibility and so become more attractive as explanations for human decision behavior.

## References

[1] Fishburn, P.C. (1974). Lexicographic orders, utilities and decision rules: A survey. *Management Science, 20,* 1442-1471.

[2] Payne, J.W., Bettman, J.R., & Johnson, E.J. (1993). *The adaptive decision maker*. New York: Cambridge University Press.

[3] Bröder, A. (2000). Assessing the empirical validity of the "Take-The-Best" heuristic as a model of human probabilistic inference. *Journal of Experimental Psychology: Learning, Memory, and Cognition, 26 (5)*, 1332-1346.

[4] Bröder, A. (2003). Decision making with the "adaptive toolbox": Influence of environmental structure, intelligence, and working memory load. *Journal of Experimental Psychology: Learning, Memory, & Cognition, 29,* 611-625.

[5] Gigerenzer, G., Todd, P.M., & The ABC Research Group (1999). *Simple heuristics that make us smart.* New York: Oxford University Press.

[6] Gigerenzer, G., & Goldstein, D.G. (1996). Reasoning the fast and frugal way: Models of bounded rationality. *Psychological Review, 103 (4)*, 650-669.

[7] Gigerenzer, G., & Goldstein, D.G. (1999). Betting on one good reason: The Take The Best Heuristic. In G. Gigerenzer, P.M. Todd & The ABC Research Group, *Simple heuristics that make us smart*. New York: Oxford University Press.

[8] Czerlinski, J., Gigerenzer, G., & Goldstein, D.G. (1999). How good are simple heuristics? In G. Gigerenzer, P.M. Todd & The ABC Research Group, *Simple heuristics that make us smart*. New York: Oxford University Press.

[9] Newell, B.R., & Shanks, D.R. (2003). Take the best or look at the rest? Factors influencing 'one-reason' decision making. *Journal of Experimental Psychology: Learning, Memory, and Cognition, 29*, 53-65.

[10] Juslin, P., & Persson, M. (2002). PROBabilities from EXemplars (PROBEX): a "lazy" algorithm for probabilistic inference from generic knowledge. *Cognitive Science, 26*, 563-607.

[11] Rivest, R. (1976). On self-organizing sequential search heuristics. *Communications of the ACM, 19(2),* 63-67.

[12] Bentley, J.L. & McGeoch, C.C. (1985). Amortized analyses of self-organizing sequential search heuristics. *Communications of the ACM, 28(4),* 404-411.

[13] Hasher, L., & Zacks, R.T. (1984). Automatic Processing of fundamental information: The case of frequency of occurrence. *American Psychologist, 39*, 1372-1388.

[14] Estes, W.K. (1976). The cognitive side of probability learning. *Psychological Review, 83*, 37-64.
